# An Improved Scheme for Detection and Labelling in Johansson Displays

**Claudio Fanti**
Computational Vision Lab, 136-93
California Institute of Technology
Pasadena, CA 91125, USA
fanti@vision.caltech.edu

**Marzia Polito**
Intel Corporation, SC12-303
2200 Mission College Blvd.
Santa Clara, CA 95054, USA
marzia.polito@intel.com

**Pietro Perona**
Computational Vision Lab, 136-93
California Institute of Technology
Pasadena, CA 91125, USA
perona@vision.caltech.edu

## Abstract

Consider a number of moving points, where each point is attached to a joint of the human body and projected onto an image plane. Johannson showed that humans can effortlessly detect and recognize the presence of other humans from such displays. This is true even when some of the body points are missing (e.g. because of occlusion) and unrelated clutter points are added to the display. We are interested in replicating this ability in a machine. To this end, we present a labelling and detection scheme in a probabilistic framework. Our method is based on representing the joint probability density of positions and velocities of body points with a graphical model, and using Loopy Belief Propagation to calculate a likely interpretation of the scene. Furthermore, we introduce a global variable representing the body's centroid. Experiments on one motion-captured sequence suggest that our scheme improves on the accuracy of a previous approach based on triangulated graphical models, especially when very few parts are visible. The improvement is due both to the more general graph structure we use and, more significantly, to the introduction of the centroid variable.

## 1 Introduction

Perceiving and analyzing human motion is a natural and useful task for our visual system. Replicating this ability in machines is one of the most important and difficult goals of machine vision. As Johannson experiments show [4], the instantaneous information on the position and velocity of a few features, such as the joints of the body, present sufficient information to detect human presence and understand the gist of human activity. This is true even if clutter features are detected in the scene,

and if some body parts features are occluded (generalized Johansson display). Selecting features in a frame, as well as computing their velocity across frames, is a task for which good quality solutions exist in the literature [5] and we will not consider it here.

We therefore assume that a number of features that are associated to the body have been detected and their velocity has been computed. We will not assume that all such features have been found, nor that all the features that were detected are associated to the body. We study the interpretation of such a generalized Johannson display, i.e. the detection of the presence of a human in the scene and the labelling of the point features as parts of the body or as clutter. We generalize an approach presented in [3] where the pattern of point positions and velocities associated to human motion was modelled with a triangulated graphical model. We are interested here in exploring the benefit of allowing long-range connections, and therefore loops in the graph representing correlations between cliques of variables. Furthermore, while [3] obtained translation invariance at the level of individual cliques, we study the possibility of obtaining translation invariance globally by introducing a variable representing the ensemble model of the body. Algorithms based on loopy belief propagation (LBP) are applied to efficiently compute high-likelihood interpretations of the scene, and therefore detection and labelling.

## 1.1 Notations

We use bold-face letters $\mathbf{x}$ for random vectors and italic letters $x$ for their sample values. The probability density (or mass) function for a variable $\mathbf{x}$ is denoted by $f_{\mathbf{x}}(x)$. When $\mathbf{x}$ is a random quantity we write the expectation as $E_{f_{\mathbf{x}}}[x]$. An ordered set $\mathcal{I} = [i_1 \ldots i_K]$ used as a vector's subscript has the obvious meaning of $\mathbf{y}_{\mathcal{I}} = [\mathbf{y}_{i_1} \ldots \mathbf{y}_{i_K}]$ or, when enclosed in squared brackets $[\mathcal{I}]_s$ applied to a dimension of a matrix $V = [v_{ij}]$, it selects the $s$-dimensional members (specified by the subscript) of the matrix along that dimension, i.e. $V_{[1:2]_4[1:2]_4}$ is the $8 \times 8$ matrix obtained by selecting the first two 4-dimensional rows and columns.

## 1.2 Problem Definition

We identify $M = 16$ relevant body parts (intuitively corresponding to the main joints). Each marked point on a display (referred to as a *detection* or *observation*) is denoted by $y_i \in \mathbb{R}^4$ and is endowed with four values, i.e. $y_i = [y_{i,a}, y_{i,b}, y_{i,v_a}, y_{i,v_b}]^T$ corresponding to its horizontal and vertical positions and velocities. Our goal here is to find the most probable assignment of a subset of detections to the body parts. For each display we call $y = [y_1^T \ldots y_N^T]^T$ the $4N \times 1$ vector of all observations (on a frame) and we model each single observation as a $4 \times 1$ random vector $\mathbf{y}_i$. In general $N \geq M$ however some or all of the $M$ parts might not be present in a given display. The binary random variable $\boldsymbol{\delta}_i$ indicates whether the $i^{th}$ part has been detected or not ($i \in \{1 \ldots M\}$) . For $i \in \{1 \ldots M\}$, a discrete random variable $\boldsymbol{\lambda}_i$ taking values on $\{1 \ldots N\}$ is used to further specify the correspondence of a body part $i$ to a particular detection $\boldsymbol{\lambda}_i$. Since this makes sense only if the body part is detected, we assume by convention that $\lambda_i = 0$ if $\delta_i = 0$. A pair $\mathbf{h} = [\boldsymbol{\lambda}, \boldsymbol{\delta}]$ is called a labelling *hypothesis*.

Any particular labelling hypothesis determines a partition of the set of indices corresponding to detections into foreground and background: $[1 \ldots N]^T = \mathcal{F} \cup \mathcal{B}$, where $\mathcal{F} = [\lambda_i : \delta_i = 1, i = 1 \ldots M]^T$ and $\mathcal{B} = [1 \ldots N]^T \setminus \mathcal{F}$. We say that $m = |\mathcal{F}|$ parts have been detected and $M - m$ are missing. Based on the partition induced on $\boldsymbol{\lambda}$ by $\boldsymbol{\delta}$, we can define two vectors $\boldsymbol{\lambda}^f = \boldsymbol{\lambda}_{\mathcal{F}}$ and $\boldsymbol{\lambda}^b = \boldsymbol{\lambda}_{\mathcal{B}}$, each identifying the detections that were assigned to the foreground and those assigned to the background

respectively. Finally, the set of detections $\mathbf{y}$ remains partitioned into the vectors $\mathbf{y}_{\boldsymbol{\lambda}^f}$ and $\mathbf{y}_{\boldsymbol{\lambda}^b}$ of the foreground and background detections respectively.

The foreground and background detections are assumed to be (conditionally) independent (given $\mathbf{h}$) meaning that their joint distribution factorizes as follows

$$f_{\mathbf{y}|\boldsymbol{\lambda}\boldsymbol{\delta}}(y|\lambda\delta) = f_{\mathbf{y}_{\boldsymbol{\lambda}^f}|\boldsymbol{\lambda}\boldsymbol{\delta}}(y_{\lambda^f}|\lambda\delta) \cdot f_{\mathbf{y}_{\boldsymbol{\lambda}^b}|\boldsymbol{\lambda}\boldsymbol{\delta}}(y_{\lambda^b}|\lambda\delta)$$

where $f_{\mathbf{y}_{\boldsymbol{\lambda}^f}|\boldsymbol{\lambda}\boldsymbol{\delta}}(y_{\lambda^f}|\lambda\delta)$ is a gaussian pdf, while $f_{\mathbf{y}_{\boldsymbol{\lambda}^b}|\boldsymbol{\lambda}\boldsymbol{\delta}}(y_{\lambda^b}|\lambda\delta)$ is the uniform pdf $\mathcal{U}_{N-m}(A)$, with $A$ determining the area of the position and velocity hyperplane for each of the $N - m$ background parts.

More specifically, when all $M$ parts are observed ($\boldsymbol{\delta} = [1 \ldots 1]^T$) we have that $f_{\mathbf{y}_{\boldsymbol{\lambda}_{[1:M]_1}}|\boldsymbol{\lambda}\boldsymbol{\delta}}(y_{\lambda_{[1:M]_1}}|\lambda\delta)$ is $\mathcal{N}(\mu, \Sigma)$. When $m \leq M$ instead, $\mathcal{N}(\mu^f, \Sigma^f)$ is the marginalized (over the $M - m$ missing parts) version $\mathcal{N}(\mu^f, \Sigma^f)$ of the complete model $\mathcal{N}(\mu, \Sigma)$.

Our goal is to find an hypothesis $\hat{h} = [\hat{\lambda}, \hat{\delta}]$ such that

$$[\hat{\lambda}, \hat{\delta}] = \arg\max_{\lambda\delta}\{Q(\lambda, \delta)\} = \arg\max_{\lambda\delta}\{f_{\mathbf{y}_{\boldsymbol{\lambda}}|\boldsymbol{\lambda}\boldsymbol{\delta}}(y_\lambda|\lambda, \delta)\}. \tag{1}$$

## 2 Learning the Model's Parameters and Structure

In this section we will assume some familiarity with the connections between probability density functions and graphical models. Let us initially assume that the moving human being we want to detect is centrally positioned in the frame. We will then enhance the model in order to accommodate for horizontal and vertical translations.

In the learning process we want to estimate the parameters of $f_{\mathbf{y}_{\boldsymbol{\lambda}^f}|\boldsymbol{\lambda}\boldsymbol{\delta}}(y_{\lambda^f}|\lambda\delta)$, where the labeling of the training set is known, $N = M$ (no clutter is present) and $\boldsymbol{\delta} = [1 \ldots 1]^T$ (all parts are visible). A fully connected graphical model would be the most accurate description of the training set, however, the search for the optimal labelling, given a display, would be computationally infeasible. Additionally, by Occam's razor, such model might not generalize as well as a simpler one. It is intuitive to think that some (conditional) independencies between the $\mathbf{y}_i$'s hold. We learn the model structure from the data, as well as the parameters. To limit the computational cost and to hope in a better generalizing model, we put an upper bound on the fan-in (number of incoming edges) of the nodes.

In order to make the trade-off between complexity and likelihood explicit, we adopt the BIC (Bayesian Information Criterion) score. We recall that the BIC score is consistent, and that since the probability distribution factorizes family-wise, the score decomposes additively. An exhaustive search among graphs is infeasible. We therefore attempt to determine the highest scoring graph by mean of a greedy hill-climbing algorithm, with random restarts. Specifically, at each step the algorithm chooses the elementary operation (among adding, removing or inverting an edge of the graph) that results in the highest increase for the score. To prevent getting stuck in local maxima, we randomly restart a number of times once we cannot get any score improvements, and then we pick the graph achieving the highest score overall. We finally obtain our model by retaining the associated maximum likelihood parameters.

As opposed to previous approaches [3], no decomposability of the graph is imposed, and exact belief propagation methods that pass through the construction of a junc-

tion tree are not applicable. When the junction property is satisfied, the maximum spanning tree algorithm allows an efficient construction of the junction tree. The tree with the most populated separators between cliques is produced in linear time. Here, we propose instead a construction of the junction graph that (greedily) attempts to minimize the complexity of the induced subgraph associated with each variable.

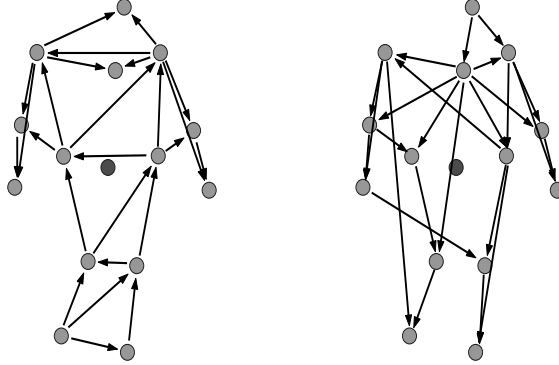

Figure 1: Graphical Models. Light shaded vertices represent variables associated to different body parts, edges indicate conditional (in)dependencies, following the standard Graphical Models conventions. [Left] Hand made decomposable graph from [3], used for comparison. [Right] Model learned from data (sequence W1, see section 4), with max fan-in constrain of 2.

## 3    Detection and Labelling with Expectation Maximization

One could solve the maximization problem (1) by means of Belief Propagation (BP), however, we require our system to be invariant with respect to translations in the first two coordinates (position) of the observations. To achieve this we introduce a new parameter $\boldsymbol{\gamma} = [\boldsymbol{\gamma}_a, \boldsymbol{\gamma}_b, 0, 0]^T$ that represents the reference system's origin, which we now allow to be different than zero. By introducing the *centered observations* $\bar{\mathbf{y}}_{\boldsymbol{\lambda}} = \mathbf{y}_{\boldsymbol{\lambda}} - \boldsymbol{\gamma}$ our model becomes

$$f_{\bar{\mathbf{y}}_{\boldsymbol{\lambda}}|\boldsymbol{\gamma}\mathbf{h}}(\bar{y}|\gamma h) = f_{\bar{\mathbf{y}}_{\boldsymbol{\lambda}^f}|\boldsymbol{\gamma}\boldsymbol{\lambda}\boldsymbol{\delta}}(\bar{y}_{\lambda^f}|\gamma\lambda\delta) \cdot f_{\bar{\mathbf{y}}_{\boldsymbol{\lambda}^b}|\boldsymbol{\lambda}\boldsymbol{\delta}}(\bar{y}_{\lambda^b}|\lambda\delta).$$

where in the second member the first factor is now $\mathcal{N}(\bar{\mu}^f, \bar{\Sigma}^f)$ while the second factor remains $\mathcal{U}_{N-m}(\bar{A})$.

We finally use an EM-like procedure to estimate $\boldsymbol{\gamma}$ obtaining, as a by-product, the maximizing hypothesis $h$ we are after.

### 3.1    E-Step

As the hypothesis $h$ is unobservable we replace the complete-data log-likelihood, with its expected value

$$\hat{L}_c(\tilde{f}, h) = E_{\tilde{f}_{\mathbf{h}}}[\log f_{\bar{\mathbf{y}}_{\boldsymbol{\lambda}}|\boldsymbol{\gamma}}(\bar{y}_{\lambda}|\gamma)] \qquad (2)$$

where the expectation is taken with respect to a generic distribution $\tilde{f}_{\mathbf{h}}(h)$. It's known that the E-step maximizing solution is $\tilde{f}_{\mathbf{h}}^{(k)}(h) \propto f_{\bar{\mathbf{y}}_{\boldsymbol{\lambda}}|\boldsymbol{\gamma}}(\bar{y}_{\lambda}|\gamma^{(k-1)})$. Since we will not be able to compute such distribution for all the assignments $h$ of $\mathbf{h}$, we will

make a so-called *hard assignment* i.e. we will approximate $f_{\bar{\mathbf{y}}_\lambda|\boldsymbol{\gamma}}(\bar{y}_\lambda|\gamma^{(k-1)})$ with $\mathbf{1}(h - h^{(k)})$, where

$$h^{(k)} = \arg\max_h\{f_{\bar{\mathbf{y}}_\lambda|\boldsymbol{\gamma}}(\bar{y}_\lambda|\gamma^{(k-1)})\}.$$

Given the current estimate $\gamma^{(k-1)}$ of $\boldsymbol{\gamma}$, the hypothesis $h^{(k)}$ can be determined by maximizing the (discrete) potential $\Pi(h) = \log f_{\bar{\mathbf{y}}_{\lambda^f}|\boldsymbol{\gamma}\mathbf{h}}(\bar{y}_{\lambda^f}|\gamma^{(k-1)}h) \cdot f_{\mathbf{y}_{\lambda^b}|\mathbf{h}}(y_{\lambda^b}|h)$ with a Max-Sum Loopy Belief Propagation (LBP) on the associated junction graph. The potential above decomposes into a number of factors (or cliques). With the exception of root nodes, each family gives rise to a factor that we initialize to the family's conditional probability mass function (pmf). For a root node, its marginal pmf is multiplied into one of its children.

If LBP converges and the determined $h^{(k)}$ maximizes the expected log-likelihood $\hat{L}_c(\tilde{f}^{(k)}, h^{(k-1)})$, then we are guaranteed (otherwise there is just reasonable[1] hope) that EM will converge to the sought-after ML estimate of $\boldsymbol{\gamma}$.

### 3.2 M-Step

In the M-Step we maximize (2) with respect to $\boldsymbol{\gamma}$, holding $h = h^{(k)}$, i.e. we compute

$$\gamma^{(k+1)} = \arg\max_\gamma\{\log f_{\bar{\mathbf{y}}_\lambda|\boldsymbol{\gamma}}(\bar{y}_{\lambda^{(k)}}|\gamma)\} \tag{3}$$

The maximizing $\gamma$ can be obtained from

$$0 = \nabla_{\boldsymbol{\gamma}}[(y_\lambda - \bar{\mu} - J\gamma)^T \bar{\Sigma}^{-1}(y_\lambda - \bar{\mu} - J\gamma)] \tag{4}$$

where $J_4 = \mathrm{diag}(1, 1, 0, 0)$ and $J = [\underbrace{\begin{array}{cccc} J_4 & J_4 & \cdots & J_4 \end{array}}_{m}]^T$.

The solution involves the inversion of the matrix $\bar{\Sigma}$ as a whole which is numerically instable given the minimal variance in the vertical component of the motion. We therefore approximate it with a block-diagonal version $\tilde{\Sigma}$ with

$$\tilde{\Sigma}_{[i]_4[i]_4} = I_4 \frac{\det(\bar{\Sigma}_{[i]_4[i]_4})}{\det(\bar{\Sigma})}. \tag{5}$$

It's easy to see that, for appropriate $\alpha_i$'s,

$$\gamma^{(k+1)} = J_4 \sum_{\delta_i=1} \left[\alpha_i(y_{\lambda_i} - \bar{\mu}_i)\right]. \tag{6}$$

### 3.3 Detection Criteria

Let $\boldsymbol{\sigma}$ be a (discrete) indicator random variable for the event that the Johansson's display represents a scene with a human body. So far, in our discussion we have implicitly assumed that $\boldsymbol{\sigma} = 1$. In the following section we will describe a way for determining whether a human body is actually present (*detection*). By defining $R(y) = \frac{f_{\boldsymbol{\sigma}|\mathbf{y}}(1|y)}{f_{\boldsymbol{\sigma}|\mathbf{y}}(0|y)}$, we claim that a human body is present whenever $R(y) > 1$. By Bayes rule, $R(y)$ can be rewritten as

$$R(y) = \frac{f_{\mathbf{y}|\boldsymbol{\sigma}}(y|1)}{f_{\mathbf{y}|\boldsymbol{\sigma}}(y|0)} \cdot \frac{f_{\boldsymbol{\sigma}}(1)}{f_{\boldsymbol{\sigma}}(0)} = \frac{f_{\mathbf{y}|\boldsymbol{\sigma}}(y|1)}{f_{\mathbf{y}|\boldsymbol{\sigma}}(y|0)} \cdot R_p$$

where $R_p = \frac{P[\boldsymbol{\sigma}=1]}{P[\boldsymbol{\sigma}=0]}$ is the contribution to $R(y)$ due to the prior on $\boldsymbol{\sigma}$. In order to compute the $R(y)$ we marginalize over the labelling hypothesis $\mathbf{h}$.

When $\boldsymbol{\sigma} = 0$, the only admissible hypotheses must have $\boldsymbol{\delta} = 0^T$ (no body parts are present) which translates into $f_{\boldsymbol{\delta}|\boldsymbol{\sigma}}(\delta|\sigma) = P[\boldsymbol{\delta} = \delta|\sigma = 0] = 1_k(\delta - 0^T)$. Also, $f_{\boldsymbol{\lambda}|\boldsymbol{\delta\sigma}}(\lambda|\delta 1) = N^{-N}$ as no labelling is more likely than any other, before we have seen the detections. All $N$ detections are labelled by $\boldsymbol{\lambda}$ as background and their conditional density is $\mathcal{U}_N(A)$. Therefore, we have $f_{\mathbf{y}|\boldsymbol{\sigma}}(y|0) = \frac{1}{A^N}\frac{1}{N^N}$ where the summation is over the $\lambda, \delta$ compatible with $\boldsymbol{\sigma} = 0$.

When $\boldsymbol{\sigma} = 1$, we have $f_{\boldsymbol{\delta}|\boldsymbol{\sigma}}(\delta|1) = P[\boldsymbol{\delta} = \delta] = 2^{-M}$ as we assume that each body part appears (or not) in a given display with probability $\frac{1}{2}$, independently of all other parts. Also, $f_{\boldsymbol{\lambda}|\boldsymbol{\delta\sigma}}(\lambda|\delta 1) = N^{-N}$ as before and therefore we can write

$$f_{\mathbf{y}|\boldsymbol{\sigma}}(y|1) = \sum_{\lambda,\delta} \left[ f_{\mathbf{y}|\boldsymbol{\lambda\delta\sigma}}(y|\lambda\delta 1) \right] \frac{1}{N^N} \frac{1}{2^M}$$

where the summation is over the $\lambda, \delta$ compatible with $\boldsymbol{\sigma} = 1$. We conclude that

$$R(y) = R_p \frac{f_{\mathbf{y}|\boldsymbol{\sigma}}(y|1)}{f_{\mathbf{y}|\boldsymbol{\sigma}}(y|0)} = R_p \frac{A^N}{2^M} \sum_{\lambda,\delta} \left[ f_{\mathbf{y}|\boldsymbol{\lambda\delta\sigma}}(y|\lambda\delta 1) \right]$$

When implementing Loopy Belief Propagation, on a finite-precision computational architecture using Gaussian models, we are unable to perform marginalization as we can only represent log-probabilities. However, we will assume that the ML labelling $\hat{h}_\sigma$ is predominant over all other labelling, so that in the estimate of $\boldsymbol{\sigma}$ we can approximate marginalization with maximization and therefore write

$$R(y) \approx R_p \frac{A^N}{2^M} f_{\mathbf{y}|\boldsymbol{\lambda\delta\sigma}}(y|\hat{\lambda}\hat{\delta}1)$$

where $\hat{\lambda}, \hat{\delta}$ is the maximizing hypothesis when $\boldsymbol{\sigma} = 1$.

## 4    Experimental Results

In our experiment we use two sequences W1 and W2[2] of about 7,000 frames each, representing a human subject walking back and forth along a straight line. Both sequences were acquired and labelled with a motion capture system. Each pair of consecutive frames is used to produce a Johannson display with positions and velocities. W1 is used to learn the probabilistic model's parameter and structure. A 700 frames random sample from W2 is then used to test of our algorithm.

We evaluate the performance of our technique and compare it with the hand-made, decomposable graphical model of [3]. There, translation invariance is achieved by using relative positions within each clique. We refer to it as to the *local* version of translation invariance (as opposed to the *global* version proposed in this paper).

We first explore the benefits of just relaxing the decomposability constrain, still implementing the translation invariance locally. The lower two dashed curves of Figure 2 already show a noticeable improvement, especially when fewer body parts are visible. However, the biggest increase in performance is brought by global translation invariance as it is evident from the upper two curves of Figure 2.

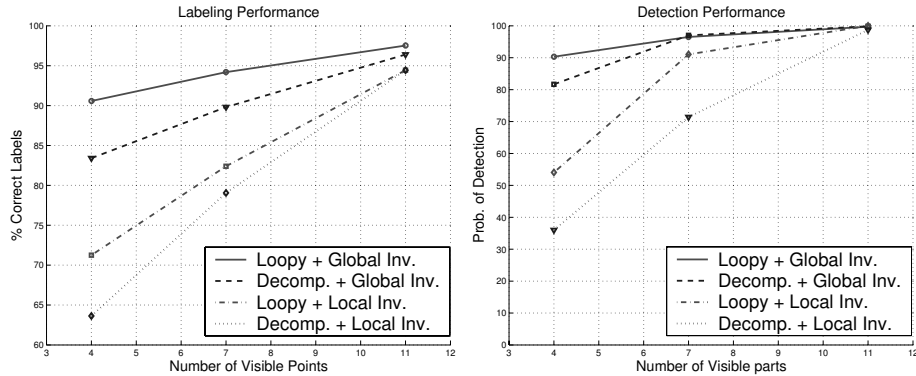

Figure 2: Detection and Labeling Performance. [Left] Labeling: On each display from the sequence W2, we randomly occlude between 3 and 10 parts and superimpose 30 randomly positioned clutter points. For any given number of visible parts, the four curves represent the percentage of correctly labeled parts out of the total labels in all 700 displays of W2. Each curve reflects a combination of either Local or Global translation invariance and Decomposable or Loopy graph. [Right] Detection: For the same four combinations we plot $P_{detection}$ (Prob. of detecting a person when the display shows one) for a fixed $P_{false-alarm} = 10\%$ (probability of stating that a person is present when only 30 points of clutters are presented). Again, we vary the number of visible points between 4, 7 and 11.

As for the dynamical programming algorithm of [3], the Loopy Belief Propagation algorithm runs in $O(MN^3)$, however 4 or 5 more iterations are needed for it to converge. Furthermore, to avoid local maxima, we restart the algorithm at most 10 times using a randomly generated schedule to pass the messages. Finally, when global invariance is used, we re-initialize $\gamma$ up to 10 times. Each time we randomly pick a value within a different region of the display. On average, about 5 restarts for $\gamma$, 5 different scheduling and 3 iterations of EM suffice to achieve a labeling with a likelihood comparable with the one of the ground truth labeling.

## 5    Discussion, Conclusions and Future Work

Generalizing our model from decomposable [3] to loopy produced a gain in performance. Further improvement would be expected when allowing larger cliques in the junction graph, at a considerable computational cost. A more sensible improvement was obtained by adding a global variable modeling the centroid of the figure.

Taking [3] as a reference, there is about a 10x increase in computational cost when we either allow a loopy graph or account for translations with the centroid. When both enhancement are present the cost increase is between 100x and 1,000x.

We believe that the combination of these two techniques points in the right direction. The local translation invariance model required the computation of relative positions within the same clique. These could not be computed in the majority of cliques when a large number of body parts were occluded, even with the more accurate loopy graphical model. Moreover, the introduction of the centroid variable is also valuable in light of a possible extension of the algorithm to multi-frame tracking.

We should also note that the structure learning technique is sub-optimal due to

the greediness of the algorithm. In addition, the model parameters and structure are estimated under the hypothesis of no occlusion or clutter. An algorithm that considers these two phenomena in the learning phase could likely achieve better results in realistic situations, when clutter and occlusion are significant.

Finally, the step towards using displays directly obtained from gray-level image sequences remains a challenge that will be the goal of future work.

### 5.1 Acknowledgements

We are very grateful to Max Welling, who first proposed the idea of using LBP to solve for the optimal labelling in a 2001 Research Note, and who gave many useful suggestion. Sequences W1 and W2 used in the experiments were collected by L. Goncalves and E. di Bernando. This work was partially funded by the NSF Center for Neuromorphic Systems Engineering grant EEC-9402726 and by the ONR MURI grant N00014-01-1-0890.

## Footnotes

[1]Experimentally it is observed that when LBP converges, the determined maximum is either global or, although local, the potential's value is very close to its global optimum. If the potential is increased (not necessarily maximized) by LBP, that suffices for EM to converge

[2]available at http://www.vision.caltech.edu/fanti.

## References

[1] Y. Song, L. Goncalves and P. Perona, "Learning Probabilistic Structure for Human Motion Detection", *Proc. IEEE Conf. Computer Vision and Pattern Recognition, vol II, pages 771-777, Kauai, Hawaii, December 2001.*

[2] Y. Song, L. Goncalves and P. Perona, "Unsupervised Learning of Human Motion Models", *Advances in Neural Information Processing Systems 14, Vancouver, Cannada, December 2001.*

[3] Y. Song, L. Goncalves, and P. Perona, "Monocular perception of biological motion - clutter and partial occlusion", *Proc. of 6th European Conferences on Computer Vision, vol II, pages 719-733, Dublin, Ireland, June/July, 2000.*

[4] G. Johansson, "Visual Perception of Biological Motion and a Model For Its Analysis", *Perception and Psychophysics 14, 201-211, 1973.*

[5] C. Tomasi and T. Kanade, "Detection and tracking of point features", *Tech. Rep. CMU-CS-91-132, Carnegie Mellon University, 1991.*

[6] S.M. Aji and R.J. McEliece, "The generalized distributive law", *IEEE Trans. Info. Theory, 46:325-343, March 2000.*

[7] P. Giudici and R Castelo, "Improving Markov Chain Monte Carlo Model Search for Data Mining", *Machine Learning 50(1-2), 127-158, 2003.*

[8] W.T.Freeman and Y. Weiss, "On the optimality of solutions of the max-product belief propagation algorithm in arbitrary graphs", *IEEE Transactions on Information Theory 47:2 pages 723-735. (2001).*

[9] J.S. Yedidia, W.T.Freeman and Y. Weiss, "Bethe free energy, Kikuchi approximations and belief propagation algorithms", *Advances in Neural Information Processing Systems 13, Vancouver, Canada, December 2000.*

[10] D. Chickering, "Optimal Structure Identification with Greedy Search", *Journal of Machine Learning Research 3, pages 507-554 (2002).*
